# People Tracking with the Laplacian Eigenmaps Latent Variable Model

**Zhengdong Lu**
CSEE, OGI, OHSU
zhengdon@csee.ogi.edu

**Miguel Á. Carreira-Perpiñán**
EECS, UC Merced
http://eecs.ucmerced.edu

**Cristian Sminchisescu**
University of Bonn
sminchisescu.ins.uni-bonn.de

## Abstract

Reliably recovering 3D human pose from monocular video requires models that bias the estimates towards typical human poses and motions. We construct priors for people tracking using the Laplacian Eigenmaps Latent Variable Model (LELVM). LELVM is a recently introduced probabilistic dimensionality reduction model that combines the advantages of latent variable models—a multimodal probability density for latent and observed variables, and globally differentiable nonlinear mappings for reconstruction and dimensionality reduction—with those of spectral manifold learning methods—no local optima, ability to unfold highly nonlinear manifolds, and good practical scaling to latent spaces of high dimension. LELVM is computationally efficient, simple to learn from sparse training data, and compatible with standard probabilistic trackers such as particle filters. We analyze the performance of a LELVM-based probabilistic sigma point mixture tracker in several real and synthetic human motion sequences and demonstrate that LELVM not only provides sufficient constraints for robust operation in the presence of missing, noisy and ambiguous image measurements, but also compares favorably with alternative trackers based on PCA or GPLVM priors.

Recent research in reconstructing articulated human motion has focused on methods that can exploit available prior knowledge on typical human poses or motions in an attempt to build more reliable algorithms. The high-dimensionality of human ambient pose space—between 30-60 joint angles or joint positions depending on the desired accuracy level, makes exhaustive search prohibitively expensive. This has negative impact on existing trackers, which are often not sufficiently reliable at reconstructing human-like poses, self-initializing or recovering from failure. Such difficulties have stimulated research in algorithms and models that reduce the effective working space, either using generic search focusing methods (annealing, state space decomposition, covariance scaling) or by exploiting specific problem structure (e.g. kinematic jumps). Experience with these procedures has nevertheless shown that any search strategy, no matter how effective, can be made significantly more reliable if restricted to low-dimensional state spaces. This permits a more thorough exploration of the typical solution space, for a given, comparatively similar computational effort as a high-dimensional method. The argument correlates well with the belief that the human pose space, although high-dimensional in its natural ambient parameterization, has a significantly lower perceptual (latent or intrinsic) dimensionality, at least in a practical sense—many poses that are possible are so improbable in many real-world situations that it pays off to encode them with low accuracy.

A perceptual representation has to be powerful enough to capture the diversity of human poses in a sufficiently broad domain of applicability (the task domain), yet compact and analytically tractable for search and optimization. This justifies the use of models that are nonlinear and low-dimensional (able to unfold highly nonlinear manifolds with low distortion), yet probabilistically motivated and globally continuous for efficient optimization. Reducing dimensionality is not the only goal: perceptual representations have to preserve critical properties of the ambient space. Reliable tracking needs locality: nearby regions in ambient space *have to* be mapped to nearby regions in latent space. If this does not hold, the tracker is forced to make unrealistically large, and difficult to predict jumps in latent space in order to follow smooth trajectories in the joint angle ambient space.

In this paper we propose to model priors for articulated motion using a recently introduced probabilistic dimensionality reduction method, the Laplacian Eigenmaps Latent Variable Model (LELVM) [1]. Section 1 discusses the requirements of priors for articulated motion in the context of probabilistic and spectral methods for manifold learning, and section 2 describes LELVM and shows how it combines both types of methods in a principled way. Section 3 describes our tracking framework (using a particle filter) and section 4 shows experiments with synthetic and real human motion sequences using LELVM priors learned from motion-capture data.

**Related work:** There is significant work in human tracking, using both generative and discriminative methods. Due to space limitations, we will focus on the more restricted class of 3D generative algorithms based on learned state priors, and not aim at a full literature review. Deriving compact prior representations for tracking people or other articulated objects is an active research field, steadily growing with the increased availability of human motion capture data. Howe et al. and Sidenbladh et al. [2] propose Gaussian mixture representations of short human motion fragments (snippets) and integrate them in a Bayesian MAP estimation framework that uses 2D human joint measurements, independently tracked by scaled prismatic models [3]. Brand [4] models the human pose manifold using a Gaussian mixture and uses an HMM to infer the mixture component index based on a temporal sequence of human silhouettes. Sidenbladh et al. [5] use similar dynamic priors and exploit ideas in texture synthesis—efficient nearest-neighbor search for similar motion fragments at runtime—in order to build a particle-filter tracker with observation model based on contour and image intensity measurements. Sminchisescu and Jepson [6] propose a low-dimensional probabilistic model based on fitting a parametric reconstruction mapping (sparse radial basis function) and a parametric latent density (Gaussian mixture) to the embedding produced with a spectral method. They track humans walking and involved in conversations using a Bayesian multiple hypotheses framework that fuses contour and intensity measurements. Urtasun et al. [7] use a dynamic MAP estimation framework based on a GPLVM and 2D human joint correspondences obtained from an independent image-based tracker. Li et al. [8] use a coordinated mixture of factor analyzers within a particle filtering framework, in order to reconstruct human motion in multiple views using chamfer matching to score different configuration. Wang et al. [9] learn a latent space with associated dynamics where both the dynamics and observation mapping are Gaussian processes, and Urtasun et al. [10] use it for tracking. Taylor et al. [11] also learn a binary latent space with dynamics (using an energy-based model) but apply it to synthesis, not tracking. Our work learns a static, generative low-dimensional model of poses and integrates it into a particle filter for tracking. We show its ability to work with real or partially missing data and to track multiple activities.

# 1  Priors for articulated human pose

We consider the problem of learning a probabilistic low-dimensional model of human articulated motion. Call $\mathbf{y} \in \mathbb{R}^D$ the representation in ambient space of the articulated pose of a person. In this paper, $\mathbf{y}$ contains the 3D locations of anywhere between 10 and 60 markers located on the person's joints (other representations such as joint angles are also possible). The values of $\mathbf{y}$ have been normalised for translation and rotation in order to remove rigid motion and leave only the articulated motion (see section 3 for how we track the rigid motion). While $\mathbf{y}$ is high-dimensional, the motion pattern lives in a low-dimensional manifold because most values of $\mathbf{y}$ yield poses that violate body constraints or are simply atypical for the motion type considered. Thus we want to model $\mathbf{y}$ in terms of a small number of latent variables $\mathbf{x}$ given a collection of poses $\{\mathbf{y}_n\}_{n=1}^N$ (recorded from a human with motion-capture technology). The model should satisfy the following: (1) It should define a probability density for $\mathbf{x}$ and $\mathbf{y}$, to be able to deal with noise (in the image or marker measurements) and uncertainty (from missing data due to occlusion or markers that drop), and to allow integration in a sequential Bayesian estimation framework. The density model should also be flexible enough to represent multimodal densities. (2) It should define mappings for dimensionality reduction $\mathbf{F}$ : $\mathbf{y} \rightarrow \mathbf{x}$ and reconstruction $\mathbf{f}$ : $\mathbf{x} \rightarrow \mathbf{y}$ that apply to any value of $\mathbf{x}$ and $\mathbf{y}$ (not just those in the training set); and such mappings should be defined on a global coordinate system, be continuous (to avoid physically impossible discontinuities) and differentiable (to allow efficient optimisation when tracking), yet flexible enough to represent the highly nonlinear manifold of articulated poses. From a statistical machine learning point of view, this is precisely what *latent variable models* (LVMs) do; for example, factor analysis defines linear mappings and Gaussian densities, while the generative topographic mapping (GTM; [12]) defines nonlinear mappings and a Gaussian-mixture density in ambient space. However, factor analysis is too limited to be of practical use, and GTM—

while flexible—has two important practical problems: (1) the latent space must be discretised to allow tractable learning and inference, which limits it to very low (2–3) latent dimensions; (2) the parameter estimation is prone to bad local optima that result in highly distorted mappings.

Another dimensionality reduction method recently introduced, GPLVM [13], which uses a Gaussian process mapping $\mathbf{f}(\mathbf{x})$, partly improves this situation by defining a tunable parameter $\mathbf{x}_n$ for each data point $\mathbf{y}_n$. While still prone to local optima, this allows the use of a better initialisation for $\{\mathbf{x}_n\}_{n=1}^N$ (obtained from a spectral method, see later). This has prompted the application of GPLVM for tracking human motion [7]. However, GPLVM has some disadvantages: its training is very costly (each step of the gradient iteration is cubic on the number of training points $N$, though approximations based on using few points exist); unlike true LVMs, it defines neither a posterior distribution $p(\mathbf{x}|\mathbf{y})$ in latent space nor a dimensionality reduction mapping $\mathrm{E}\{\mathbf{x}|\mathbf{y}\}$; and the latent representation it obtains is not ideal. For example, for periodic motions such as running or walking, repeated periods (identical up to small noise) can be mapped apart from each other in latent space because nothing constrains $\mathbf{x}_n$ and $\mathbf{x}_m$ to be close even when $\mathbf{y}_n = \mathbf{y}_m$ (see fig. 3 and [10]).

There exists a different type of dimensionality reduction methods, *spectral methods* (such as Isomap, LLE or Laplacian eigenmaps [14]), that have advantages and disadvantages complementary to those of LVMs. They define neither mappings nor densities but just a correspondence $(\mathbf{x}_n, \mathbf{y}_n)$ between points in latent space $\mathbf{x}_n$ and ambient space $\mathbf{y}_n$. However, the training is efficient (a sparse eigenvalue problem) and has no local optima, and often yields a correspondence that successfully models highly nonlinear, convoluted manifolds such as the Swiss roll. While these attractive properties have spurred recent research in spectral methods, their lack of mappings and densities has limited their applicability in people tracking. However, a new model that combines the advantages of LVMs and spectral methods in a principled way has been recently proposed [1], which we briefly describe next.

## 2   The Laplacian Eigenmaps Latent Variable Model (LELVM)

LELVM is based on a natural way of defining an out-of-sample mapping for Laplacian eigenmaps (LE) which, in addition, results in a density model. In LE, typically we first define a $k$-nearest-neighbour graph on the sample data $\{\mathbf{y}_n\}_{n=1}^N$ and weigh each edge $\mathbf{y}_n \sim \mathbf{y}_m$ by a Gaussian affinity function $K(\mathbf{y}_n, \mathbf{y}_m) = w_{nm} = \exp\left(-\frac{1}{2}\left\|(\mathbf{y}_n - \mathbf{y}_m)/\sigma\right\|^2\right)$. Then the latent points $\mathbf{X}$ result from:

$$\min \operatorname{tr}\left(\mathbf{X}\mathbf{L}\mathbf{X}^\top\right) \quad \text{s.t.} \quad \mathbf{X} \in \mathbb{R}^{L \times N}, \ \mathbf{X}\mathbf{D}\mathbf{X}^\top = \mathbf{I}, \ \mathbf{X}\mathbf{D}\mathbf{1} = \mathbf{0} \tag{1}$$

where we define the matrix $\mathbf{X}_{L \times N} = (\mathbf{x}_1, \ldots, \mathbf{x}_N)$, the symmetric affinity matrix $\mathbf{W}_{N \times N}$, the degree matrix $\mathbf{D} = \operatorname{diag}\left(\sum_{n=1}^N w_{nm}\right)$, the graph Laplacian matrix $\mathbf{L} = \mathbf{D} - \mathbf{W}$, and $\mathbf{1} = (1, \ldots, 1)^\top$. The constraints eliminate the two trivial solutions $\mathbf{X} = \mathbf{0}$ (by fixing an arbitrary scale) and $\mathbf{x}_1 = \cdots = \mathbf{x}_N$ (by removing $\mathbf{1}$, which is an eigenvector of $\mathbf{L}$ associated with a zero eigenvalue). The solution is given by the leading $\mathbf{u}_2, \ldots, \mathbf{u}_{L+1}$ eigenvectors of the normalised affinity matrix $\mathbf{N} = \mathbf{D}^{-\frac{1}{2}}\mathbf{W}\mathbf{D}^{-\frac{1}{2}}$, namely $\mathbf{X} = \mathbf{V}^\top \mathbf{D}^{-\frac{1}{2}}$ with $\mathbf{V}_{N \times L} = (\mathbf{v}_2, \ldots, \mathbf{v}_{L+1})$ (an a posteriori translated, rotated or uniformly scaled $\mathbf{X}$ is equally valid).

Following [1], we now define an out-of-sample mapping $\mathbf{F}(\mathbf{y}) = \mathbf{x}$ for a new point $\mathbf{y}$ as a semi-supervised learning problem, by recomputing the embedding as in (1) (i.e., augmenting the graph Laplacian with the new point), but keeping the old embedding fixed:

$$\min_{\mathbf{x} \in \mathbb{R}^L} \operatorname{tr}\left( \begin{pmatrix} \mathbf{X} & \mathbf{x} \end{pmatrix} \begin{pmatrix} \mathbf{L} & \mathbf{K}(\mathbf{y}) \\ \mathbf{K}(\mathbf{y})^\top & \mathbf{1}^\top\mathbf{K}(\mathbf{y}) \end{pmatrix} \begin{pmatrix} \mathbf{X}^\top \\ \mathbf{x}^\top \end{pmatrix} \right) \tag{2}$$

where $K_n(\mathbf{y}) = K(\mathbf{y}, \mathbf{y}_n) = \exp\left(-\frac{1}{2}\left\|(\mathbf{y} - \mathbf{y}_n)/\sigma\right\|^2\right)$ for $n = 1, \ldots, N$ is the kernel induced by the Gaussian affinity (applied only to the $k$ nearest neighbours of $\mathbf{y}$, i.e., $K_n(\mathbf{y}) = 0$ if $\mathbf{y} \not\sim \mathbf{y}_n$). This is one natural way of adding a new point to the embedding by keeping existing embedded points fixed. We need not use the constraints from (1) because they would trivially determine $\mathbf{x}$, and the uninteresting solutions $\mathbf{X} = \mathbf{0}$ and $\mathbf{X} = \text{constant}$ were already removed in the old embedding anyway. The solution yields an out-of-sample dimensionality reduction mapping $\mathbf{x} = \mathbf{F}(\mathbf{y})$:

$$\mathbf{x} = \mathbf{F}(\mathbf{y}) = \frac{\mathbf{X}\,\mathbf{K}(\mathbf{y})}{\mathbf{1}^\top\mathbf{K}(\mathbf{y})} = \sum_{n=1}^N \frac{K(\mathbf{y}, \mathbf{y}_n)}{\sum_{n'=1}^N K(\mathbf{y}, \mathbf{y}_{n'})} \mathbf{x}_n \tag{3}$$

applicable to any point $\mathbf{y}$ (new or old). This mapping is formally identical to a Nadaraya-Watson estimator (kernel regression; [15]) using as data $\{(\mathbf{x}_n, \mathbf{y}_n)\}_{n=1}^N$ and the kernel $K$. We can take this a step further by defining a LVM that has as joint distribution a kernel density estimate (KDE):

$$p(\mathbf{x}, \mathbf{y}) = \frac{1}{N} \sum_{n=1}^N K_\mathbf{y}(\mathbf{y}, \mathbf{y}_n) K_\mathbf{x}(\mathbf{x}, \mathbf{x}_n) \quad p(\mathbf{y}) = \frac{1}{N} \sum_{n=1}^N K_\mathbf{y}(\mathbf{y}, \mathbf{y}_n) \quad p(\mathbf{x}) = \frac{1}{N} \sum_{n=1}^N K_\mathbf{x}(\mathbf{x}, \mathbf{x}_n)$$

where $K_{\mathbf{y}}$ is proportional to $K$ so it integrates to 1, and $K_{\mathbf{x}}$ is a pdf kernel in $\mathbf{x}$–space. Consequently, the marginals in observed and latent space are also KDEs, and the dimensionality reduction and reconstruction mappings are given by kernel regression (the conditional means $\mathrm{E}\left\{\mathbf{y}|\mathbf{x}\right\}$, $\mathrm{E}\left\{\mathbf{x}|\mathbf{y}\right\}$):

$$\mathbf{F}(\mathbf{y}) = \sum_{n=1}^{N} p(n|\mathbf{y})\mathbf{x}_n \qquad \mathbf{f}(\mathbf{x}) = \sum_{n=1}^{N} \frac{K_{\mathbf{x}}(\mathbf{x},\mathbf{x}_n)}{\sum_{n'=1}^{N} K_{\mathbf{x}}(\mathbf{x},\mathbf{x}_{n'})}\mathbf{y}_n = \sum_{n=1}^{N} p(n|\mathbf{x})\mathbf{y}_n. \qquad (4)$$

We allow the bandwidths to be different in the latent and ambient spaces: $K_{\mathbf{x}}(\mathbf{x},\mathbf{x}_n) \propto \exp\left(-\frac{1}{2}\left\|(\mathbf{x}-\mathbf{x}_n)/\sigma_{\mathbf{x}}\right\|^2\right)$ and $K_{\mathbf{y}}(\mathbf{y},\mathbf{y}_n) \propto \exp\left(-\frac{1}{2}\left\|(\mathbf{y}-\mathbf{y}_n)/\sigma_{\mathbf{y}}\right\|^2\right)$. They may be tuned to control the smoothness of the mappings and densities [1].

Thus, LELVM naturally extends a LE embedding (efficiently obtained as a sparse eigenvalue problem with a cost $\mathcal{O}(N^2)$) to global, continuous, differentiable mappings (NW estimators) and potentially multimodal densities having the form of a Gaussian KDE. This allows easy computation of posterior probabilities such as $p(\mathbf{x}|\mathbf{y})$ (unlike GPLVM). It can use a continuous latent space of arbitrary dimension $L$ (unlike GTM) by simply choosing $L$ eigenvectors in the LE embedding. It has no local optima since it is based on the LE embedding. LELVM can learn convoluted mappings (e.g. the Swiss roll) and define maps and densities for them [1]. The only parameters to set are the graph parameters (number of neighbours $k$, affinity width $\sigma$) and the smoothing bandwidths $\sigma_{\mathbf{x}}$, $\sigma_{\mathbf{y}}$.

## 3  Tracking framework

We follow the sequential Bayesian estimation framework, where for state variables $\mathbf{s}$ and observation variables $\mathbf{z}$ we have the recursive prediction and correction equations:

$$p(\mathbf{s}_t|\mathbf{z}_{0:t-1}) = \int p(\mathbf{s}_t|\mathbf{s}_{t-1})\, p(\mathbf{s}_{t-1}|\mathbf{z}_{0:t-1})\, d\mathbf{s}_{t-1} \qquad p(\mathbf{s}_t|\mathbf{z}_{0:t}) \propto p(\mathbf{z}_t|\mathbf{s}_t)\, p(\mathbf{s}_t|\mathbf{z}_{0:t-1}). \qquad (5)$$

We define the state variables as $\mathbf{s} = (\mathbf{x},\mathbf{d})$ where $\mathbf{x} \in \mathbb{R}^L$ is the low-dim. latent space (for pose) and $\mathbf{d} \in \mathbb{R}^3$ is the centre-of-mass location of the body (in the experiments our state also includes the orientation of the body, but for simplicity here we describe only the translation). The observed variables $\mathbf{z}$ consist of image features or the perspective projection of the markers on the camera plane. The mapping from state to observations is (for the markers' case, assuming $M$ markers):

$$\begin{aligned}\mathbf{x} \in \mathbb{R}^L \xrightarrow{\ \mathbf{f}\ } \mathbf{y} \in \mathbb{R}^{3M} &\longrightarrow \oplus \xrightarrow{\ P\ } \mathbf{z} \in \mathbb{R}^{2M}\\ \mathbf{d} \in \mathbb{R}^3 &\longrightarrow \end{aligned} \qquad (6)$$

where $\mathbf{f}$ is the LELVM reconstruction mapping (learnt from mocap data); $\oplus$ shifts each 3D marker by $\mathbf{d}$; and $P$ is the perspective projection (pinhole camera), applied to each 3D point separately. Here we use a simple observation model $p(\mathbf{z}_t|\mathbf{s}_t)$: Gaussian with mean given by the transformation (6) and isotropic covariance (set by the user to control the influence of measurements in the tracking). We assume known correspondences and observations that are obtained either from the 3D markers (for tracking synthetic data) or 2D tracks obtained from a 2D tracker. Our dynamics model is

$$p(\mathbf{s}_t|\mathbf{s}_{t-1}) \propto p_{\mathbf{d}}(\mathbf{d}_t|\mathbf{d}_{t-1})\, p_{\mathbf{x}}(\mathbf{x}_t|\mathbf{x}_{t-1})\, p(\mathbf{x}_t) \qquad (7)$$

where both dynamics models for $\mathbf{d}$ and $\mathbf{x}$ are random walks: Gaussians centred at the previous step value $\mathbf{d}_{t-1}$ and $\mathbf{x}_{t-1}$, respectively, with isotropic covariance (set by the user to control the influence of dynamics in the tracking); and $p(\mathbf{x}_t)$ is the LELVM prior. Thus the overall dynamics predicts states that are both near the previous state and yield feasible poses. Of course, more complex dynamics models could be used if e.g. the speed and direction of movement are known.

As tracker we use the Gaussian mixture Sigma-point particle filter (GMSPPF) [16]. This is a particle filter that uses a Gaussian mixture representation for the posterior distribution in state space and updates it with a Sigma-point Kalman filter. This Gaussian mixture will be used as proposal distribution to draw the particles. As in other particle filter implementations, the prediction step is carried out by approximating the integral (5) with particles and updating the particles' weights. Then, a new Gaussian mixture is fitted with a weighted EM algorithm to these particles. This replaces the resampling stage needed by many particle filters and mitigates the problem of sample depletion while also preventing the number of components in the Gaussian mixture from growing over time. The choice of this particular tracker is not critical; we use it to illustrate the fact that LELVM can be introduced in any probabilistic tracker for nonlinear, nongaussian models. Given the corrected distribution $p(\mathbf{s}_t|\mathbf{z}_{0:t})$, we choose its mean as recovered state (pose and location). It is also possible to choose instead the mode closest to the state at $t-1$, which could be found by mean-shift or Newton algorithms [17] since we are using a Gaussian-mixture representation in state space.

# 4 Experiments

We demonstrate our low-dimensional tracker on image sequences of people walking and running, both synthetic (fig. 1) and real (fig. 2–3). Fig. 1 shows the model copes well with persistent partial occlusion and severely subsampled training data (**A**,**B**), and quantitatively evaluates temporal reconstruction (**C**). For all our experiments, the LELVM parameters (number of neighbors $k$, Gaussian affinity $\sigma$, and bandwidths $\sigma_{\mathbf{x}}$ and $\sigma_{\mathbf{y}}$) were set manually. We mainly considered 2D latent spaces (for pose, plus 6D for rigid motion), which were expressive enough for our experiments. More complex, higher-dimensional models are straightforward to construct. The initial state distribution $p(\mathbf{s}_0)$ was chosen a broad Gaussian, the dynamics and observation covariance were set manually to control the tracking smoothness, and the GMSPPF tracker used a 5-component Gaussian mixture in latent space (and in the state space of rigid motion) and a small set of 500 particles. The 3D representation we use is a 102-D vector obtained by concatenating the 3D markers coordinates of all the body joints. These would be highly unconstrained if estimated independently, but we only use them as intermediate representation; tracking actually occurs in the latent space, tightly controlled using the LELVM prior. For the synthetic experiments and some of the real experiments (figs. 2–3) the camera parameters and the body proportions were known (for the latter, we used the 2D outputs of [6]). For the CMU mocap video (fig. 2**B**) we roughly guessed. We used mocap data from several sources (CMU, OSU). As observations we always use 2D marker positions, which, depending on the analyzed sequence were either known (the synthetic case), or provided by an existing tracker [6] or specified manually (fig. 2**B**). Alternatively 2D point trackers similar to the ones of [7] can be used. The forward generative model is obtained by combining the latent to ambient space mapping (this provides the position of the 3D markers) with a perspective projection transformation. The observation model is a product of Gaussians, each measuring the probability of a particular marker position given its corresponding image point track.

**Experiments with synthetic data:** we analyze the performance of our tracker in controlled conditions (noise perturbed synthetically generated image tracks) both under regular circumstances (reasonable sampling of training data) and more severe conditions with subsampled training points and persistent partial occlusion (the man running behind a fence, with many of the 2D marker tracks obstructed). Fig. 1**B**,**C** shows both the posterior (filtered) latent space distribution obtained from our tracker, and its mean (we do not show the distribution of the global rigid body motion; in all experiments this is tracked with good accuracy). In the latent space plot shown in fig. 1**B**, the onset of running (two cycles were used) appears as a separate region external to the main loop. It does not appear in the subsampled training set in fig. 1**B**, where only one running cycle was used for training and the onset of running was removed. In each case, one can see that the model is able to track quite competently, with a modest decrease in its temporal accuracy, shown in fig. 1**C**, where the averages are computed per 3D joint (normalised wrt body height). Subsampling causes some ambiguity in the estimate, e.g. see the bimodality in the right plot in fig. 1**C**. In another set of experiments (not shown) we also tracked using different subsets of 3D markers. The estimates were accurate even when about 30% of the markers were dropped.

**Experiments with real images:** this shows our tracker's ability to work with real motions of different people, with different body proportions, *not* in its latent variable model training set (figs. 2–3). We study walking, running and turns. In all cases, tracking and 3D reconstruction are reasonably accurate. We have also run comparisons against low-dimensional models based on PCA and GPLVM (fig. 3). It is important to note that, for LELVM, errors in the pose estimates are primarily caused by mismatches between the mocap data used to learn the LELVM prior and the body proportions of the person in the video. For example, the body proportions of the OSU motion captured walker are quite different from those of the image in fig. 2–3 (e.g. note how the legs of the stick man are shorter relative to the trunk). Likewise, the style of the runner from the OSU data (e.g. the swinging of the arms) is quite different from that of the video. Finally, the interest points tracked by the 2D tracker do not entirely correspond either in number or location to the motion capture markers, and are noisy and sometimes missing. In future work, we plan to include an optimization step to also estimate the body proportions. This would be complicated for a general, unconstrained model because the dimensions of the body couple with the pose, so either one or the other can be changed to improve the tracking error (the observation likelihood can also become singular). But for dedicated prior pose models like ours these difficulties should be significantly reduced. The model simply cannot assume highly unlikely stances—these are either not representable at all, or have reduced probability—and thus avoids compensatory, unrealistic body proportion estimates.

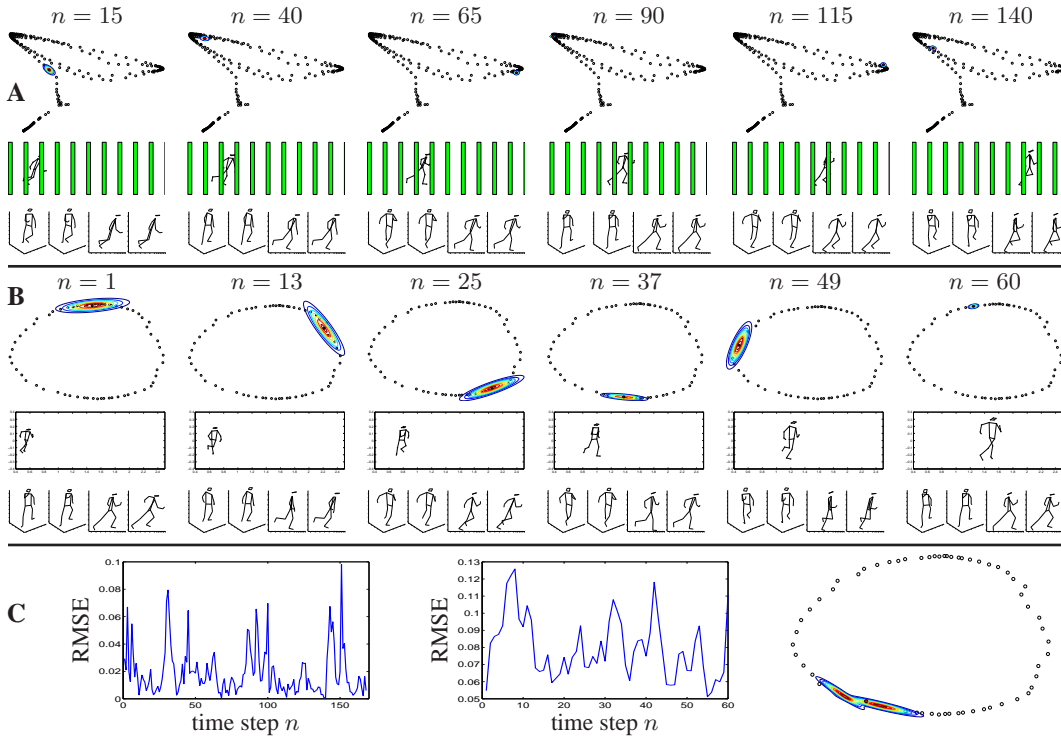

Figure 1: OSU running man motion capture data. **A**: we use 217 datapoints for training LELVM (with added noise) and for tracking. *Row 1*: tracking in the 2D latent space. The contours (very tight in this sequence) are the posterior probability. *Row 2*: perspective-projection-based observations with occlusions. *Row 3*: each quadruplet $(a, a', b, b')$ show the true pose of the running man from a front and side views $(a, b)$, and the reconstructed pose by tracking with our model $(a', b')$. **B**: we use the first running cycle for training LELVM and the second cycle for tracking. **C**: RMSE errors for each frame, for the tracking of **A** (*left plot*) and **B** (*middle plot*), normalised so that 1 equals the height of the stick man. $\text{RMSE}(n) = \left(\frac{1}{M}\sum_{j=1}^{M}\|\mathbf{y}_{nj} - \hat{\mathbf{y}}_{nj}\|^2\right)^{-1/2}$ for all 3D locations of the $M$ markers, i.e., comparison of reconstructed stick man $\hat{\mathbf{y}}_n$ with ground-truth stick man $\mathbf{y}_n$. *Right plot*: multimodal posterior distribution in pose space for the model of **A** (frame 42).

**Comparison with PCA and GPLVM** (fig. 3): for these models, the tracker uses the same GMSPPF setting as for LELVM (number of particles, initialisation, random-walk dynamics, etc.) but with the mapping $\mathbf{y} = \mathbf{f}(\mathbf{x})$ provided by GPLVM or PCA, and with a uniform prior $p(\mathbf{x})$ in latent space (since neither GPLVM nor the non-probabilistic PCA provide one). The LELVM-tracker uses both its $\mathbf{f}(\mathbf{x})$ and latent space prior $p(\mathbf{x})$, as discussed. All methods use a 2D latent space. We ensured the best possible training of GPLVM by model selection based on multiple runs. For PCA, the latent space looks deceptively good, showing non-intersecting loops. However, (1) individual loops do not collect together as they should (for LELVM they do); (2) worse still, the mapping from 2D to pose space yields a poor observation model. The reason is that the loop in 102-D pose space is nonlinearly bent and a plane can at best intersect it at a few points, so the tracker often stays put at one of those (typically an "average" standing position), since leaving it would increase the error a lot. Using more latent dimensions would improve this, but as LELVM shows, this is not necessary. For GPLVM, we found high sensitivity to filter initialisation: the estimates have high variance across runs and are inaccurate $\approx 80\%$ of the time. When it fails, the GPLVM tracker often freezes in latent space, like PCA. When it does succeed, it produces results that are comparable with LELVM, although somewhat less accurate visually. However, even then GPLVM's latent space consists of continuous chunks spread apart and offset from each other; GPLVM has no incentive to place nearby two $\mathbf{x}$s mapping to the same $\mathbf{y}$. This effect, combined with the lack of a data-sensitive, realistic latent space density $p(\mathbf{x})$, makes GPLVM jump erratically from chunk to chunk, in contrast with LELVM, which smoothly follows the 1D loop. Some GPLVM problems might be alleviated using higher-order dynamics, but our experiments suggest that such modeling sophistication is less

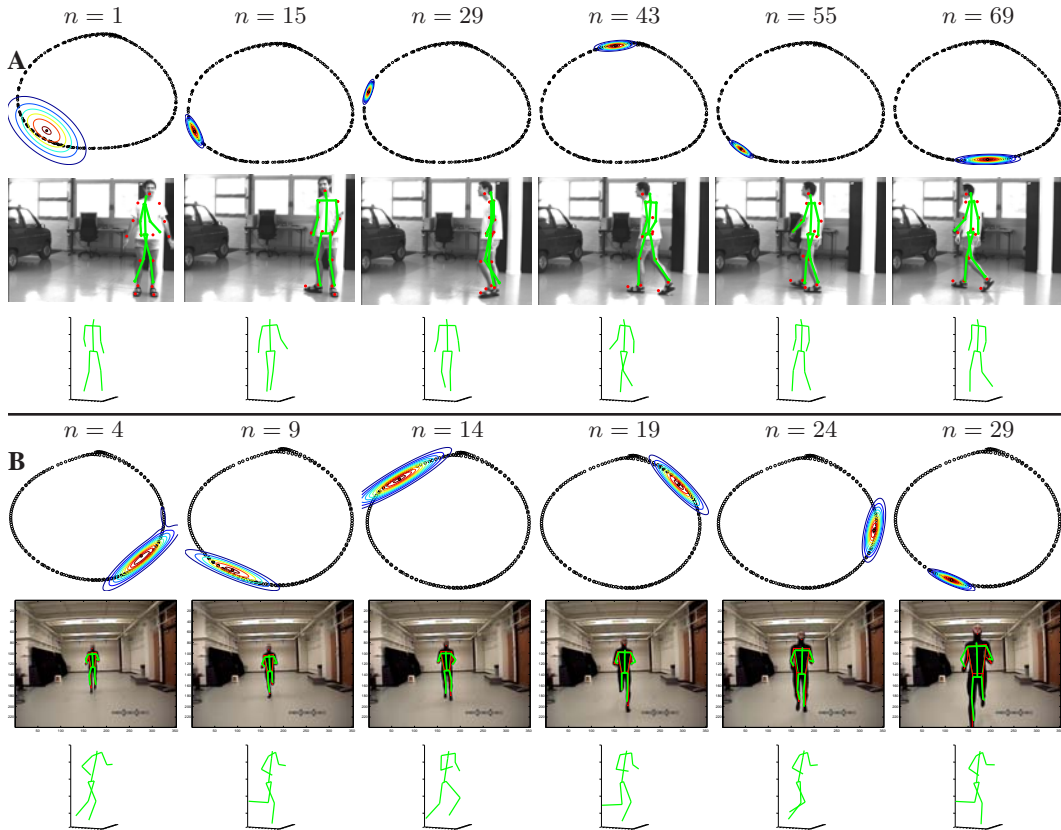

Figure 2: **A**: tracking of a video from [6] (turning & walking). We use 220 datapoints (3 full walking cycles) for training LELVM. *Row 1*: tracking in the 2D latent space. The contours are the estimated posterior probability. *Row 2*: tracking based on markers. The red dots are the 2D tracks and the green stick man is the 3D reconstruction obtained using our model. *Row 3*: our 3D reconstruction from a different viewpoint. **B**: tracking of a person running straight towards the camera. Notice the scale changes and possible forward-backward ambiguities in the 3D estimates. We train the LELVM using 180 datapoints (2.5 running cycles); 2D tracks were obtained by manually marking the video. In both **A**–**B** the mocap training data was for a person different from the video's (with different body proportions and motions), and no ground-truth estimate was available for favourable initialisation.

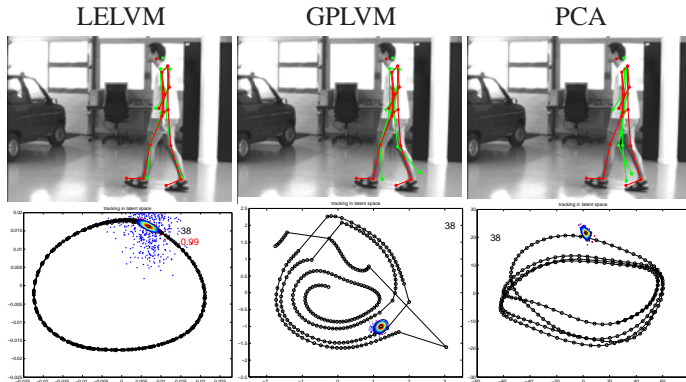

Figure 3: Method comparison, frame 38. PCA and GPLVM map consecutive walking cycles to spatially distinct latent space regions. Compounded by a data independent latent prior, the resulting tracker gets easily confused: it jumps across loops and/or remains put, trapped in local optima. In contrast, LELVM is stable and follows tightly a 1D manifold (see videos).

crucial if locality constraints are correctly modeled (as in LELVM). We conclude that, compared to LELVM, GPLVM is significantly less robust for tracking, has much higher training overhead and lacks some operations (e.g. computing latent conditionals based on partly missing ambient data).

# 5 Conclusion and future work

We have proposed the use of priors based on the Laplacian Eigenmaps Latent Variable Model (LELVM) for people tracking. LELVM is a probabilistic dim. red. method that combines the advantages of latent variable models and spectral manifold learning algorithms: a multimodal probability density over latent and ambient variables, globally differentiable nonlinear mappings for reconstruction and dimensionality reduction, no local optima, ability to unfold highly nonlinear manifolds, and good practical scaling to latent spaces of high dimension. LELVM is computationally efficient, simple to learn from sparse training data, and compatible with standard probabilistic trackers such as particle filters. Our results using a LELVM-based probabilistic sigma point mixture tracker with several real and synthetic human motion sequences show that LELVM provides sufficient constraints for robust operation in the presence of missing, noisy and ambiguous image measurements. Comparisons with PCA and GPLVM show LELVM is superior in terms of accuracy, robustness and computation time. The objective of this paper was to demonstrate the ability of the LELVM prior in a simple setting using 2D tracks obtained automatically or manually, and single-type motions (running, walking). Future work will explore more complex observation models such as silhouettes; the combination of different motion types in the same latent space (whose dimension will exceed 2); and the exploration of multimodal posterior distributions in latent space caused by ambiguities.

## Acknowledgments

This work was partially supported by NSF CAREER award IIS–0546857 (MACP), NSF IIS–0535140 and EC MCEXT–025481 (CS). CMU data: `http://mocap.cs.cmu.edu` (created with funding from NSF EIA–0196217). OSU data: `http://accad.osu.edu/research/mocap/mocap_data.htm`.

## References

[1] M. Á. Carreira-Perpiñán and Z. Lu. The Laplacian Eigenmaps Latent Variable Model. In *AISTATS*, 2007.

[2] N. R. Howe, M. E. Leventon, and W. T. Freeman. Bayesian reconstruction of 3D human motion from single-camera video. In *NIPS*, volume 12, pages 820–826, 2000.

[3] T.-J. Cham and J. M. Rehg. A multiple hypothesis approach to figure tracking. In *CVPR*, 1999.

[4] M. Brand. Shadow puppetry. In *ICCV*, pages 1237–1244, 1999.

[5] H. Sidenbladh, M. J. Black, and L. Sigal. Implicit probabilistic models of human motion for synthesis and tracking. In *ECCV*, volume 1, pages 784–800, 2002.

[6] C. Sminchisescu and A. Jepson. Generative modeling for continuous non-linearly embedded visual inference. In *ICML*, pages 759–766, 2004.

[7] R. Urtasun, D. J. Fleet, A. Hertzmann, and P. Fua. Priors for people tracking from small training sets. In *ICCV*, pages 403–410, 2005.

[8] R. Li, M.-H. Yang, S. Sclaroff, and T.-P. Tian. Monocular tracking of 3D human motion with a coordinated mixture of factor analyzers. In *ECCV*, volume 2, pages 137–150, 2006.

[9] J. M. Wang, D. Fleet, and A. Hertzmann. Gaussian process dynamical models. In *NIPS*, volume 18, 2006.

[10] R. Urtasun, D. J. Fleet, and P. Fua. Gaussian process dynamical models for 3D people tracking. In *CVPR*, pages 238–245, 2006.

[11] G. W. Taylor, G. E. Hinton, and S. Roweis. Modeling human motion using binary latent variables. In *NIPS*, volume 19, 2007.

[12] C. M. Bishop, M. Svensén, and C. K. I. Williams. GTM: The generative topographic mapping. *Neural Computation*, 10(1):215–234, January 1998.

[13] N. Lawrence. Probabilistic non-linear principal component analysis with Gaussian process latent variable models. *Journal of Machine Learning Research*, 6:1783–1816, November 2005.

[14] M. Belkin and P. Niyogi. Laplacian eigenmaps for dimensionality reduction and data representation. *Neural Computation*, 15(6):1373–1396, June 2003.

[15] B. W. Silverman. *Density Estimation for Statistics and Data Analysis*. Chapman & Hall, 1986.

[16] R. van der Merwe and E. A. Wan. Gaussian mixture sigma-point particle filters for sequential probabilistic inference in dynamic state-space models. In *ICASSP*, volume 6, pages 701–704, 2003.

[17] M. Á. Carreira-Perpiñán. Acceleration strategies for Gaussian mean-shift image segmentation. In *CVPR*, pages 1160–1167, 2006.

